# Confidence Intervals for the Area under the ROC Curve

**Corinna Cortes**
Google Research
1440 Broadway
New York, NY 10018
corinna@google.com

**Mehryar Mohri**
Courant Institute, NYU
719 Broadway
New York, NY 10003
mohri@cs.nyu.edu

## Abstract

In many applications, good ranking is a highly desirable performance for a classifier. The criterion commonly used to measure the ranking quality of a classification algorithm is the area under the ROC curve (AUC). To report it properly, it is crucial to determine an interval of confidence for its value. This paper provides confidence intervals for the AUC based on a statistical and combinatorial analysis using only simple parameters such as the error rate and the number of positive and negative examples. The analysis is distribution-independent, it makes no assumption about the distribution of the scores of negative or positive examples. The results are of practical use and can be viewed as the equivalent for AUC of the standard confidence intervals given in the case of the error rate. They are compared with previous approaches in several standard classification tasks demonstrating the benefits of our analysis.

## 1 Motivation

In many machine learning applications, the ranking quality of a classifier is critical. For example, the ordering of the list of relevant documents returned by a search engine or a document classification system is essential. The criterion widely used to measure the ranking quality of a classification algorithm is the area under an ROC curve (AUC). But, to measure and report the AUC properly, it is crucial to determine an interval of confidence for its value as it is customary for the error rate and other measures. It is also important to make the computation of the confidence interval practical by relying only on a small and simple number of parameters. In the case of the error rate, such intervals are often derived from just the sample size $N$.

We present an extensive theoretical analysis of the AUC and show that a similar confidence interval can be derived for its value using only simple parameters such as the error rate $k/N$, the number of positive examples $m$, and the number of negative examples $n = N - m$. Thus, our results extend to AUC the computation of confidence intervals from a small number of readily available parameters.

Our analysis is distribution-independent in the sense that it makes no assumption about the distribution of the scores of negative or positive examples. The use of the error rate helps determine tight confidence intervals. This contrasts with existing approaches presented in the statistical literature [11, 5, 2] which are based either on weak distribution-independent assumptions resulting in too loose confidence intervals, or strong distribution-dependent assumptions leading to tight but unsafe confidence intervals.

We show that our results are of practical use. We also compare them with previous approaches in several standard classification tasks demonstrating the benefits of our analysis. Our results are also useful for testing the statistical significance of the difference of the AUC values of two classifiers.

The paper is organized as follows. We first introduce the definition of the AUC, its connection with the Wilcoxon-Mann-Whitney statistic (Section 2), and briefly review some essential aspects of the existing literature related to the computation of confidence intervals for the AUC. Our computation of the expected value and variance of the AUC for a fixed error rate requires establishing several combinatorial identities. Section 4 presents some existing identities and gives the proof of novel ones useful for the computation of the variance. Section 5 gives the reduced expressions for the expected value and variance of the AUC for a fixed error rate. These can be efficiently computed and used to determine our confidence intervals for the AUC (Section 6). Section 7 reports the result of the comparison of our method with previous approaches, including empirical results for several standard tasks.

## 2 Definition and Properties of the AUC

The *Receiver Operating Characteristics* (ROC) curves were originally introduced in signal detection theory [6] in connection with the study of radio signals, and have been used since then in many other applications, in particular for medical decision-making. Over the last few years, they have found increased interest in the machine learning and data mining communities for model evaluation and selection [14, 13, 7, 12, 16, 3]. The ROC curve for a binary classification problem plots the true positive rate as a function of the false positive rate. The points of the curve are obtained by sweeping the classification threshold from the most positive classification value to the most negative. For a fully random classification, the ROC curve is a straight line connecting the origin to $(1,1)$. Any improvement over random classification results in an ROC curve at least partially above this straight line. The AUC is defined as the area under the ROC curve.

Consider a binary classification task with $m$ positive examples and $n$ negative examples. Let $C$ be a fixed classifier that outputs a strictly ordered list for these examples. Let $x_1, \ldots, x_m$ be the output of $C$ on the positive examples and $y_1, \ldots, y_n$ its output on the negative examples and denote by $1_X$ the indicator function of a set $X$. Then, the AUC, A, associated to $C$ is given by:

$$A = \frac{\sum_{i=1}^{m} \sum_{j=1}^{n} 1_{x_i > y_j}}{mn} \tag{1}$$

which is the value of the *Wilcoxon-Mann-Whitney statistic* [10]. Thus, the AUC is closely related to the ranking quality of the classification. It can be viewed as a measure based on pairwise comparisons between classifications of the two classes. It is an estimate of the probability $P_{xy}$ that the classifier ranks a randomly chosen positive example higher than a negative example. With a perfect ranking, all positive examples are ranked higher than the negative ones and $A = 1$. Any deviation from this ranking decreases the AUC, and the expected AUC value for a random ranking is 0.5.

## 3 Overview of Related Work

This section briefly describes some previous distribution-dependent approaches presented in the statistical literature to derive confidence intervals for the AUC and compares them to our method. The starting point for these analyses is a formula giving the variance of the AUC, $A$, for a fixed distribution of the scores $\mathbf{P}_x$ of the positive examples and $\mathbf{P}_y$ of the negative examples [10, 1]:

$$\sigma_A^2 = \frac{A(1 - A) + (m - 1)(\mathbf{P}_{xxy} - A^2) + (n - 1)(\mathbf{P}_{xyy} - A^2)}{mn} \tag{2}$$

where $\mathbf{P}_{xxy}$ is the probability that the classifier ranks two randomly chosen positive examples higher than a negative one, and $\mathbf{P}_{xyy}$ the probability that it ranks two randomly chosen negative examples lower than a positive one. To compute the variance exactly using Equation 2, the distributions $\mathbf{P}_x$ and $\mathbf{P}_y$ must be known.

Hanley and McNeil [10] argue in favor of exponential distributions, loosely claiming that this upper-bounds the variance of normal distributions with various means and ratios of variances. They show that for exponential distributions $\mathbf{P}_{xxy} = \frac{A}{2-A}$ and $\mathbf{P}_{xyy} = \frac{2A^2}{1+A}$. The resulting confidence intervals are of course relatively tight, but their validity is questionable since they are based on a strong assumption about the distributions of the positive and negative scores that may not hold in many cases.

An alternative considered by several authors to the exact computation of the variance is to determine instead the maximum of the variance over all possible continuous distributions with the same expected value of the AUC. For all such distributions, one can fix $m$ and $n$ and compute the expected AUC and its variance. The maximum variance is denoted by $\sigma^2_{max}$ and is given by [5, 2]:

$$\sigma^2_{max} = \frac{A(1-A)}{\min\{m,n\}} \leq \frac{1}{4\min\{m,n\}} \tag{3}$$

Unfortunately, this often yields loose confidence intervals of limited practical use.

Our approach for computing the mean and variance of the AUC is distribution-independent and inspired by the machine learning literature where analyses typically center on the error rate. We require only that the error rate be measured and compute the mean and variance of the AUC over all distributions $\mathbf{P}_x$ and $\mathbf{P}_y$ that maintain the same error rate. Our approach is in line with that of [5, 2] but it crucially avoids considering the maximum of the variance. We show that it is possible to compute directly the mean and variance of the AUC assigning equal weight to all the possible distributions. Of course, one could argue that not all distributions $\mathbf{P}_x$ and $\mathbf{P}_y$ are equally probable, but since these distributions are highly problem-dependent, we find it risky to make any general assumption on the distributions and thereby limit the validity of our results. Our approach is further justified empirically by the experiments reported in the last section.

## 4 Combinatorial Analysis

The analysis of the statistical properties of the AUC given a fixed error rate requires various combinatorial calculations. This section describes several of the combinatorial identities that are used in our computation of the confidence intervals. For all $q \geq 0$, let $X_q(k,m,n)$ be defined by:

$$X_q(k,m,n) = \sum_{x=0}^{k} x^q \binom{M}{x} \binom{M'}{x'} \tag{4}$$

where $M = m - (k-x) + x$, $M' = n + (k-x) - x$, and $x' = k - x$. In previous work, we derived the following two identities which we used to compute the expected value of the AUC [4]:

$$X_0(k,m,n) = \sum_{x=0}^{k} \binom{n+m+1}{x} \quad X_1(k,m,n) = \sum_{x=0}^{k} \frac{(k-x)(m-n)+k}{2} \binom{n+m+1}{x}$$

To simplify the expression of the variance of the AUC, we need to compute $X_2(k,m,n)$.

**Proposition 1** *Let $k,m,n$ be non-negative integers such that $k \leq \min\{m,n\}$, then:*

$$X_2(k,m,n) = \sum_{x=0}^{k} P_2(k,m,n,x) \binom{m+n+1}{x} \tag{5}$$

*where $P_2$ is the following 4th-degree polynomial:* $P_2(k, m, n, x) = (k - x)/12(-2x^3 + 2x^2(2m - n + 2k - 4) + x(-3m^2 + 3nm + 3m - 5km - 2k^2 + 2 + k + nk + 6n) + (3(k - 1)m^2 - 3nm(k - 1) + 6km + 5m + k^2m + 8n + 8 - 9nk + 3k + k^2 + k^2n)).$

*Proof.* The proof of the proposition is left to a longer version of this paper. $\square$

## 5 Expectation and Variance of the AUC

This section presents the expression of the expectation and variance of the AUC for a fixed error rate $k/N$ assuming that all classifications or rankings with $k$ errors are equiprobable. For a given classification, there may be $x, 0 \leq x \leq k$, false positive examples. Since the number of errors is fixed, there are $x' = k - x$ false negative examples. The expression $X_q$ discussed in the previous section represents the $q$-th moment of $x$ over all classifications with exactly $k$ errors. In previous work, we gave the exact expression of the expectation of the AUC for a fixed number of errors $k$:

**Proposition 2 ([4])** *Assume that a binary classification task with $m$ positive examples and $n$ negative examples is given. Then, the expected value of the AUC, A, over all classifications with $k$ errors is given by:*

$$\mathbf{E}[A] = 1 - \frac{k}{m + n} - \frac{(n - m)^2(m + n + 1)}{4mn} \left( \frac{k}{m + n} - \frac{\sum_{x=0}^{k-1} \binom{m+n}{x}}{\sum_{x=0}^{k} \binom{m+n+1}{x}} \right).$$

Note that the two sums in this expression cannot be further simplified since they are known not to admit a closed form [9]. We also gave the expression of the variance of the AUC in terms of the function $F$ defined for all $Y$ by:

$$F(Y) = \frac{\sum_{x=0}^{k} \binom{M}{x}\binom{M'}{x'} Y}{\sum_{x=0}^{k} \binom{M}{x}\binom{M'}{x'}}. \tag{6}$$

The following proposition reproduces that result:

**Proposition 3 ([4])** *Assume that a binary classification task with $m$ positive examples and $n$ negative examples is given. Then, the variance of the AUC A over all classifications with $k$ errors is given by:* $\sigma^2(A) = F((1 - \frac{\frac{x}{n} + \frac{k-x}{m}}{2})^2) - F((1 - \frac{\frac{x}{n} + \frac{k-x}{m}}{2}))^2 + F(\frac{mx^2 + n(k-x)^2 + (m(m+1)x + n(n+1)(k-x)) - 2x(k-x)(m+n+1)}{12m^2n^2}).$

Because of the products of binomial terms, the computation of the variance using this expression is inefficient even for relatively small values of $m$ and $n$. This expression can however be reduced using the identities presented in the previous section which leads to significantly more efficient computations that we have been using in all our experiments.

**Corollary 1 ([4])** *Assume that a binary classification task with $m$ positive examples and $n$ negative examples is given. Then, the variance of the AUC A over all classifications with $k$ errors is given by:* $\sigma^2(A) = \frac{(m+n+1)(m+n)(m+n-1)T((m+n-2)Z_4 - (2m-n+3k-10)Z_3)}{72m^2n^2} + \frac{(m+n+1)(m+n)T(m^2-nm+3km-5m+2k^2-nk+12-9k)Z_2}{48m^2n^2} - \frac{(m+n+1)^2(m-n)^4Z_1^2}{16m^2n^2} - \frac{(m+n+1)Q_1Z_1}{72m^2n^2} + \frac{kQ_0}{144m^2n^2}$ *with:*
$Z_i = \frac{\sum_{x=0}^{k-i} \binom{m+n+1-i}{x}}{\sum_{x=0}^{k} \binom{m+n+1}{x}}, T = 3((m - n)^2 + m + n) + 2,$ *and:*

$Q_0 = (m + n + 1)Tk^2 + ((-3n^2 + 3mn + 3m + 1)T - 12(3mn + m + n) - 8)k + (-3m^2 + 7m + 10n + 3nm + 10)T - 4(3mn + m + n + 1)$

$Q_1 = Tk^3 + 3(m - 1)Tk^2 + ((-3n^2 + 3mn - 3m + 8)T - 6(6mn + m + n))k + (-3m^2 + 7(m + n) + 3mn)T - 2(6mn + m + n)$

*Proof.* The expression of the variance given in Proposition 3 requires the computation of $X_q(k, m, n)$, $q = 0, 1, 2$. Using the identities giving the expressions of $X_0$ and $X_1$ and Proposition 1, which provides the expression of $X_2$, $\sigma^2(A)$ can be reduced to the expression given by the corollary. $\qquad\square$

## 6   Theory and Analysis

Our estimate of the confidence interval for the AUC is based on a simple and natural assumption. The main idea for its computation is the following. Assume that a confidence interval $E = [e_1, e_2]$ is given for the error rate of a classifier $C$ over a sample $S$, with the confidence level $1 - \epsilon$. This interval may have have been derived from a binomial model of $C$, which is a standard assumption for determining a confidence interval for the error rate, or from any other model used to compute that interval. For a given error rate $e \in E$, or equivalently for a given number of misclassifications, we can use the expectation and variance computed in the previous section and Chebyshev's inequality to predict a confidence interval $A_e$ for the AUC at the confidence level $1 - \epsilon'$. Since our equiprobable model for the classifications is independent of the model used to compute the interval of confidence for the error rate, we can use $E$ and $A_e$, $e \in E$, to compute a confidence interval of the AUC at the level $(1 - \epsilon)(1 - \epsilon')$.

**Theorem 1** *Let $C$ be a binary classifier and let $S$ be a data sample of size $N$ with $m$ positive examples and $n$ negative examples, $N = m + n$. Let $E = [e_1, e_2]$ be a confidence interval for the error rate of $C$ over $S$ at the confidence level $1 - \epsilon$. Then, for any $\epsilon'$, $0 \le \epsilon' \le 1$, we can compute a confidence interval for the AUC value of the classifier $C$ at the confidence level $(1 - \epsilon)(1 - \epsilon')$ that depends only on $\epsilon$, $\epsilon'$, $m$, $n$, and the interval $E$.*

*Proof.* Let $k_1 = Ne_1$ and $k_2 = Ne_2$ be the number of errors associated to the error rates $e_1$ and $e_2$, and let $I_K$ be the interval $I_K = [k_1, k_2]$. For a fixed $k \in I_K$, by Propositions 2 and Corollary 1, we can compute the exact value of the expectation $\mathbf{E}[A_k]$ and variance $\sigma^2(A_k)$ of the AUC $A_k$. Using Chebyshev's inequality, for any $k \in I_K$ and any $\epsilon_k > 0$,

$$\mathbf{P}\left( |A_k - \mathbf{E}[A_k]| \ge \frac{\sigma(A_k)}{\sqrt{\epsilon_k}} \right) \le \epsilon_k \tag{7}$$

where $\mathbf{E}[A_k]$ and $\sigma(A_k)$ are the expressions given in Propositions 2 and Corollary 1, which depend only on $k$, $m$, and $n$. Let $\alpha_1$ and $\alpha_2$ be defined by:

$$\alpha_1 = \min_{k \in I_K} \left\{ \mathbf{E}[A_k] - \frac{\sigma(A_k)}{\sqrt{\epsilon_k}} \right\} \qquad \alpha_2 = \max_{k \in I_K} \left\{ \mathbf{E}[A_k] + \frac{\sigma(A_k)}{\sqrt{\epsilon_k}} \right\} \tag{8}$$

$\alpha_1$ and $\alpha_2$ only depend on $I_K$ (i.e., on $e_1$ and $e_2$), and on $k$, $m$, and $n$. Let $I_A$ be the confidence interval defined by $I_A = [\alpha_1, \alpha_2]$ and let $\epsilon_k = \epsilon'$ for all $k \in I_K$. Using the fact that the confidence interval $E$ is independent of our equiprobability model for fixed-$k$ AUC values and the Bayes' rule:

$$\mathbf{P}(A \in I_A) = \sum_{k \in \mathbb{R}_+} P(A \in I_A \mid K = k)P(K = k) \tag{9}$$

$$\ge \sum_{k \in I_K} P(A \in I_A \mid K = k)P(K = k) \tag{10}$$

$$\ge (1 - \epsilon') \sum_{k \in I_K} P(K = k) \ge (1 - \epsilon')(1 - \epsilon) \tag{11}$$

where we used the property of Eq. 7 and the definitions of the intervals $I_K$ and $I_A$. Thus, $I_A$ constitutes a confidence interval for the AUC value of $C$ at the confidence level $(1 - \epsilon)(1 - \epsilon')$. $\qquad\square$

In practice, the confidence interval $E$ is often determined as a result of the assumption that $C$ follows a binomial law. This leads to the following theorem.

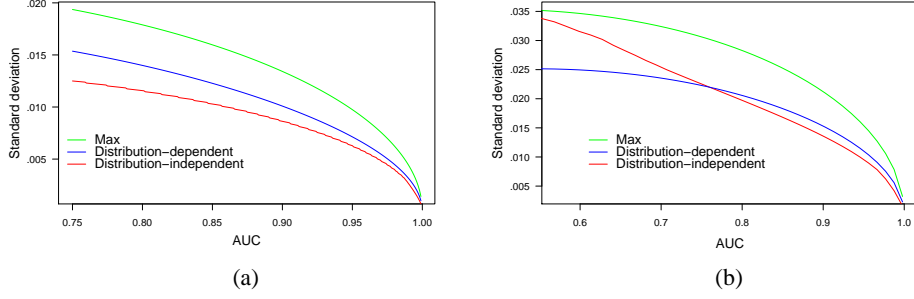

(a)                                                        (b)

Figure 1: Comparison of the standard deviations for three different methods with: (a) $m = n = 500$; (b) $m = 400$ and $n = 200$. The curves are obtained by computing the expected AUC and its standard deviations for different values of the error rate using the maximum-variance approach (Eq. 3), our distribution-independent method, and the distribution-dependent approach of Hanley [10].

**Theorem 2** *Let $C$ be a binary classifier, let $S$ be a data sample of size $N$ with $m$ positive examples and $n$ negative examples, $N = m + n$, and let $k_0$ be the number of misclassifications of $C$ on $S$. Assume that $C$ follows a binomial law, then, for any $\epsilon$, $0 \leq \epsilon \leq 1$, we can compute a confidence interval of the AUC value of the classifier $C$ at the confidence level $1 - \epsilon$ that depends only on $\epsilon$, $k_0$, $m$, and $n$.*

*Proof.* Assume that $C$ follows a binomial law with coefficient $p$. Then, Chebyshev's inequality yields:

$$\mathbf{P}(|C - \mathbf{E}[C]| \geq \eta) \leq \frac{p(1-p)}{N\eta^2} \leq \frac{1}{4N\eta^2} \tag{12}$$

Thus, $E = [\frac{k_0}{N} - \frac{1}{2\sqrt{(1-\sqrt{1-\epsilon})N}}, \frac{k_0}{N} + \frac{1}{2\sqrt{(1-\sqrt{1-\epsilon})N}}]$ forms a confidence interval for the error rate of $C$ at the confidence level $\sqrt{1-\epsilon}$. By Theorem 1, we can compute for the AUC value a confidence interval at the level $(1 - (1 - \sqrt{1-\epsilon}))(1 - (1 - \sqrt{1-\epsilon})) = 1 - \epsilon$ depending only on $\epsilon$, $m$, $n$, and the interval $E$, i.e., $k_0$, $N = m + n$, and $\epsilon$. $\square$

For large $N$, we can use the normal approximation of the binomial law to determine a finer interval $E$. Indeed, for large $N$,

$$\mathbf{P}(|C - \mathbf{E}[C]| \geq \eta) \leq 2\Phi(2\sqrt{N}\eta) \tag{13}$$

with $\Phi(u) = \int_u^\infty \frac{e^{-x^2/2}}{\sqrt{2\pi}}dx$. Thus, $E = [\frac{k_0}{N} - \frac{\Phi^{-1}(\frac{1-\sqrt{1-\epsilon}}{2})}{2\sqrt{N}}, \frac{k_0}{N} + \frac{\Phi^{-1}(\frac{1-\sqrt{1-\epsilon}}{2})}{2\sqrt{N}}]$ is the confidence interval for the error rate at the confidence level $\sqrt{1-\epsilon}$.

For simplicity, in the proof of Theorem 2, $\epsilon_k$ was chosen to be a constant ($\epsilon_k = \epsilon'$) but, in general, it can be another function of $k$ leading to tighter confidence intervals. The results presented in the next section were obtained with $\epsilon_k = a_0 \exp((k - k_0)^2/2a_1^2)$, where $a_0$ and $a_1$ are constants selected so that the inequality 11 be verified.

## 7 Experiments and Comparisons

The analysis in the previous section provides a principled method for computing a confidence interval of the AUC value of a classier $C$ at the confidence level $1 - \epsilon$ that depends only on $k$, $n$ and $m$. As already discussed, other expressions found in the statistical literature lead to either too loose or unsafely narrow confidence intervals based on questionable assumptions on the probability functions $\mathbf{P}_x$ and $\mathbf{P}_y$ [10, 15]. Figure 1 shows a comparison of the standard deviations obtained using the maximum-approach (Eq. 3), the distribution-dependent expression from [10], and our distribution-independent method for

| NAME | $m+n$ | $\frac{n}{m+n}$ | AUC | $\frac{k}{m+n}$ | $\sigma_{\text{indep}}$ | $\sigma_{\text{A}}$ | $\sigma_{\text{dep}}$ | $\sigma_{\text{max}}$ |
|---|---|---|---|---|---|---|---|---|
| pima | 368 | 0.63 | 0.70 | 0.24 | 0.0297 | 0.0440 | 0.0269 | 0.0392 |
| yeast | 700 | 0.67 | 0.63 | 0.26 | 0.0277 | 0.0330 | 0.0215 | 0.0317 |
| credit | 303 | 0.54 | 0.87 | 0.13 | 0.0176 | 0.0309 | 0.0202 | 0.0281 |
| internet-ads | 1159 | 0.17 | 0.85 | 0.05 | 0.0177 | 0.0161 | 0.0176 | 0.0253 |
| page-blocks | 2473 | 0.10 | 0.84 | 0.03 | 0.0164 | 0.0088 | 0.0161 | 0.0234 |
| ionosphere | 201 | 0.37 | 0.85 | 0.13 | 0.0271 | 0.0463 | 0.0306 | 0.0417 |

Table 1: Accuracy and AUC values for AdaBoost [8] and estimated standard deviations for several datasets from the UC Irvine repository. $\sigma_{\text{indep}}$ is a distribution-independent standard deviation obtained using our method (Theorem 2). $\sigma_{\text{A}}$ is given by Eq. (2) with the values of $A$, $\mathbf{P}_{xxy}$, and $\mathbf{P}_{xyy}$ derived from data. $\sigma_{\text{dep}}$ is the distribution-dependent standard deviation of Hanley [10], which is based on assumptions that may not always hold. $\sigma_{\text{max}}$ is defined by Eq. (3). All results were obtained on a randomly selected test set of size $m+n$.

various error rates. For $m = n = 500$, our distribution-independent method consistently leads to tighter confidence intervals (Fig. 1 (a)). It also leads to tighter confidence intervals for AUC values more than .75 for the uneven distribution $m = 400$ and $n = 200$ (Fig. 1 (b)). For lower AUC values, the distribution-dependent approach produces tighter intervals, but its underlying assumptions may not hold.

A different comparison was made using several datasets available from the UC Irvine repository (Table 1). The table shows that our estimates of the standard deviations ($\sigma_{\text{indep}}$) are in general close to or tighter than the distribution-dependent standard deviation $\sigma_{\text{dep}}$ of Hanley [10]. This is despite we do not make any assumption about the distributions of positive and negative examples. In contrast, Hanley's method is based on specific assumptions about these distributions. Plots of the actual ranking distribution demonstrate that these assumptions are often violated however. Thus, the relatively good performance of Hanley's approach on several data sets can be viewed as fortuitous and is not general. Our distribution-independent method provides tight confidence intervals, in some cases tighter than those derived from $\sigma_A$, in particular because it exploits the information provided by the error rate. Our analysis can also be used to determine if the AUC values produced by two classifiers are statistically significant by checking if the AUC value of one falls within the confidence interval of the other.

## 8    Conclusion

We presented principled techniques for computing useful confidence intervals for the AUC from simple parameters: the error rate, and the negative and positive sample sizes. We demonstrated the practicality of these confidence intervals by comparing them to previous approaches in several tasks. We also derived the exact expression of the variance of the AUC for a fixed $k$, which can be of interest in other analyses related to the AUC.

The Wilcoxon-Mann-Whitney statistic is a general measure of the quality of a ranking that is an estimate of the probability that the classifier ranks a randomly chosen positive example higher than a negative example. One could argue that accuracy at the top or the bottom of the ranking is of higher importance. This, however, contrarily to some belief, is already captured to a certain degree by the definition of the Wilcoxon-Mann-Whitney statistic which penalizes *more* errors at the top or the bottom of the ranking. It is however an interesting research problem to determine how to incorporate this bias in a stricter way in the form of a score-specific weight in the ranking measure, a weighted Wilcoxon-Mann-Whitney statistic, or how to compute the corresponding expected value and standard deviation in a general way and design machine learning algorithms to optimize such a mea-

sure. A preliminary analysis suggests, however, that the calculation of the expectation and the variance are likely to be extremely complex in that case. Finally, it could also be interesting but difficult to adapt our results to the distribution-dependent case and compare them to those of [10].

## Acknowledgments

We thank Rob Schapire for pointing out to us the problem of the statistical significance of the AUC, Daryl Pregibon for the reference to [11], and Saharon Rosset for various discussions about the topic of this paper.

## References

[1] D. Bamber. The Area above the Ordinal Dominance Graph and the Area below the Receiver Operating Characteristic Graph. *Journal of Math. Psychology*, 12, 1975.

[2] Z. W. Birnbaum and O. M. Klose. Bounds for the Variance of the Mann-Whitney Statistic. *Annals of Mathematical Statistics*, 38, 1957.

[3] J-H. Chauchat, R. Rakotomalala, M. Carloz, and C. Pelletier. Targeting Customer Groups using Gain and Cost Matrix; a Marketing Application. Technical report, ERIC Laboratory - University of Lyon 2, 2001.

[4] Corinna Cortes and Mehryar Mohri. AUC Optimization vs. Error Rate Minimization. In *Advances in Neural Information Processing Systems (NIPS 2003)*, volume 16, Vancouver, Canada, 2004. MIT Press.

[5] D. Van Dantzig. On the Consistency and Power of Wilcoxon's Two Sample Test. In *Koninklijke Nederlandse Akademie van Weterschappen, Series A*, volume 54, 1915.

[6] J. P. Egan. *Signal Detection Theory and ROC Analysis*. Academic Press, 1975.

[7] C. Ferri, P. Flach, and J. Hernández-Orallo. Learning Decision Trees Using the Area Under the ROC Curve. In *Proceedings of the 19th International Conference on Machine Learning*. Morgan Kaufmann, 2002.

[8] Yoav Freund and Robert E. Schapire. A Decision Theoretical Generalization of On-Line Learning and an Application to Boosting. In *Proceedings of the Second European Conference on Computational Learning Theory*, volume 2, 1995.

[9] Ronald L. Graham, Donald E. Knuth, and Oren Patashnik. *Concrete Mathematics*. Addison-Wesley, Reading, Massachusetts, 1994.

[10] J. A. Hanley and B. J. McNeil. The Meaning and Use of the Area under a Receiver Operating Characteristic (ROC) Curve. *Radiology*, 1982.

[11] E. L. Lehmann. *Nonparametrics: Statistical Methods Based on Ranks*. Holden-Day, San Francisco, California, 1975.

[12] M. C. Mozer, R. Dodier, M. D. Colagrosso, C. Guerra-Salcedo, and R. Wolniewicz. Prodding the ROC Curve: Constrained Optimization of Classifier Performance. In *Neural Information Processing Systems (NIPS 2002)*. MIT Press, 2002.

[13] C. Perlich, F. Provost, and J. Simonoff. Tree Induction vs. Logistic Regression: A Learning Curve Analysis. *Journal of Machine Learning Research*, 2003.

[14] F. Provost and T. Fawcett. Analysis and Visualization of Classifier Performance: Comparison under Imprecise Class and Cost Distribution. In *Proceedings of the Third International Conference on Knowledge Discovery and Data Mining*. AAAI, 1997.

[15] Saharon Rosset. Ranking-Methods for Flexible Evaluation and Efficient Comparison of 2-Class Models. Master's thesis, Tel-Aviv University, 1999.

[16] L. Yan, R. Dodier, M. C. Mozer, and R. Wolniewicz. Optimizing Classifier Performance via the Wilcoxon-Mann-Whitney Statistics. In *Proceedings of the International Conference on Machine Learning*, 2003.
